# A Maximum Entropy Approach To Collaborative Filtering in Dynamic, Sparse, High-Dimensional Domains

**Dmitry Y. Pavlov**
NEC Laboratories America
4 Independence Way
Princeton, NJ 08540,
dpavlov@nec-labs.com

**David M. Pennock**
Overture Services, Inc.
74 N. Pasadena Ave., 3rd floor
Pasadena, CA 91103,
david.pennock@overture.com

## Abstract

We develop a maximum entropy (maxent) approach to generating recommendations in the context of a user's current navigation stream, suitable for environments where data is sparse, high-dimensional, and dynamic—conditions typical of many recommendation applications. We address sparsity and dimensionality reduction by first clustering items based on user access patterns so as to attempt to minimize the apriori probability that recommendations will cross cluster boundaries and then recommending only within clusters. We address the inherent dynamic nature of the problem by explicitly modeling the data as a time series; we show how this representational expressivity fits naturally into a maxent framework. We conduct experiments on data from ResearchIndex, a popular online repository of over 470,000 computer science documents. We show that our maxent formulation outperforms several competing algorithms in offline tests simulating the recommendation of documents to ResearchIndex users.

## 1 Introduction

*Recommender systems* attempt to automate the process of "word of mouth" recommendations within a community. Typical application environments are dynamic in many respects: users come and go, users preferences and goals change, items are added and removed, and user navigation itself is a dynamic process. Recommendation domains are also often high dimensional and sparse, with tens or hundreds of thousands of items, among which very few are known to any particular user.

Consider, for instance, the problem of generating recommendations within ResearchIndex (a.k.a., CiteSeer),[1] an online digital library of computer science papers, receiving thousands of user accesses per hour. The site automatically locates computer science papers found on the Web, indexes their full text, allows browsing via the literature citation graph, and isolates the text around citations, among other services [8]. The archive contains over 470,000

documents including the full text of each document, citation links between documents, and a wealth of user access data. With so many documents, and only seven accesses per user on average, the user-document data matrix is exceedingly sparse and thus challenging to model. In this paper, we work with the ResearchIndex data, since it is an interesting application domain, and is typical of many recommendation application areas [14].

There are two conceptually different ways of making recommendations. A *content filtering* approach is to recommend solely based on the features of a document $D$ (e.g., showing documents written by the same author(s), or textually similar documents to $D$). These methods have been shown to be good predictors [3]. Another possibility is to perform *collaborative filtering* [13] by assessing the similarities between the documents requested by the current user and the users who interacted with ResearchIndex in the past. Once the users with browsing histories similar to that of a given user are identified, an assumption is made that the future browsing patterns will be similar as well, and the prediction is made accordingly. Common measures of similarity between users include Pearson correlation coefficient [13], mean squared error [16], and vector similarity [1]. More recent work includes application of statistical machine learning techniques, such as Bayesian networks [1], dependency networks [6], singular value decomposition [14] and latent class models [7, 12]. Most of these recommendation algorithms are context and order independent: that is, the rank of recommendations does not depend on the context of the user's current navigation or on recency effects (past viewed items receive as much weight as recently viewed items).

Currently, ResearchIndex mostly employs fairly simple content-based recommenders. Our objective was to design a superior (or at least complementary) model-based recommendation algorithm that (1) is tuned for a particular user at hand, and (2) takes into account the identity of the currently-viewed document $D$, so as not the lead the user too far astray from his or her current search goal.

To overcome the sparsity and high dimensionality of the data, we cluster the documents with an objective of maximizing the likelihood that recommendable items co-occur in the same cluster. By marrying the clustering technique with the end goal of recommendation, our approach appears to do a good job at maintaining high recall (sensitivity). Similar ideas in the context of maxent were proposed recently by Goodman in [5].

We explicitly model time: each user is associated with a set of sessions, and each session is modeled as a time sequence of document accesses. We present a maxent model that effectively estimates the probability of the next visited document ID (DID) given the most recently visited DID (*"bigrams"*) and past indicative DIDs (*"triggers"*). To our knowledge, this is the first application of maxent for collaborative filtering, and one of the few published formulations that makes accurate recommendations in the context of a dynamic user session [3, 15]. We perform offline empirical tests of our recommender and compare it to competing models. The comparison shows our method is quite accurate, outperforming several other less-expressive models.

The rest of the paper is organized as follows. In Section 2, we describe the log data from ResearchIndex and how we preprocessed it. Section 3 presents the greedy algorithm for clustering the documents and discusses how the clustering helps to decompose the original prediction task. In Section 4, we give a high-level description of our maxent model and the features we used for its learning. Experimental results and comparisons with other models are discussed in Section 5. In Section 6, we draw conclusions and describe directions for future work.

## 2  Preprocessing the ResearchIndex data

Each document indexed in ResearchIndex is assigned a unique document ID (DID). Whenever a user accesses the site with a cookie-enabled browser, (s)he is identified as a new or returning user and all activity is recorded on the server side with a unique user ID (UID) and a time stamp (TID). We obtained a log file that recorded approximately 3 month worth of ResearchIndex data that can roughly be viewed as a series of requests $< TID, UID, DID >$.

In the first processing step, we aggregated the requests by the $UID$ and broke them into sessions. For a fixed UID, a session is defined as a sequence of document requests, with no two consecutive requests more than $T$ seconds apart. In our experiments we chose $T = 300$, so that if a user was inactive for more than 300 seconds, his next request was considered to mark a start of a new session.

The next processing step included heuristics, such as identifying and discarding the sessions belonging to robots (they obviously contaminate the browsing patterns of human users), collapsing all same consecutive DID accesses into a single instance of this DID (our objective was to predict what interests the user beyond the currently requested document), getting rid of all DIDs that occurred less than two times in the log (for two or fewer occurrences, it is hard to reliably train the system to predict them and evaluate performance), and finally discarding sessions containing only one document.

## 3  Dimensionality Reduction Via Clustering

Even after the log is processed, the data still remains high-dimensional (62,240 documents), and sparse, and hence still hard to model. To solve these problems we clustered the documents. Since our objective was to predict the instantaneous user interests, among many possibilities of performing the clustering we chose to cluster based on user navigation patterns.

We scanned the processed log once and for each document $D^{prev}$ accumulated the number of times the document $D^{next}$ was requested immediately after $D^{prev}$; in other words, we computed the first-order Markov statistics or bigrams. Based on the user navigation patterns encoded in bigrams, the greedy clustering is done as shown in the following pseudocode:

```
 Input: Bigrams B[i,j]; Number of Clusters C;
Output: Set S of C Clusters.
Algorithm:
0.    n_C = 0;
1.    set n = argmax_{i,j} B[i,j]                        // max number of transitions
2.    for all docs i, j such that B[i,j] == n do         // all docs with n transitions
3.        if (i.assigned == -1 and j.assigned == -1 and n_C < C)
4.            S[n_C].push(i);
5.            S[n_C].push(j);
6.            i.assigned = j.assigned = n_C;
7.            n_C + +;                                    // new cluster for i and j
8.        else if (i.assigned != -1 and j.assigned == -1)
9.            S[i.assigned].push(j);
10.           j.assigned = i.assigned;                    // j goes to cluster of i
11.       else if (i.assigned == -1 and j.assigned != -1)
12.           S[j.assigned].push(i);
13.           i.assigned = j.assigned;                    // i goes to cluster of j
14.       end if
15.       B[i,j] = -1;
16.   end for
```

Table 1: Top features for some of the clusters.

| Cluster 1 | *agent, agents, behavior, good, autonomous, ...* |
|---|---|
| Cluster 2 | *training, clustering, distance, classification, kernel, svm, ...* |
| Cluster 3 | *Web, documents, query, the_Web, queries, pages, ...* |
| Cluster 4 | *packet, fast, routing, address, the_network, ip, packets, ...* |
| Cluster 5 | *transform, channel, coding, rate_compression, images, ...* |
| Cluster 6 | *detection, agents, security, intrusion_detection, ...* |
| Cluster 7 | *traffic, rate, packet, long, wide_scheduling, ...* |
| Cluster 8 | *mobile, wireless, protocol, service, services, ...* |

17.  if $(n \geq 2)$ goto 1
18.  Return S

The algorithm starts with empty clusters and then cycles through all documents picking the pairs of documents that have the current highest joint visitation frequency as prompted by a bigram frequency (lines 1 and 2). If both documents in the selected pair are unassigned, a new cluster is allocated for them (lines 3 through 7). If one of the documents in the selected pair has been assigned to one of the previous clusters, the second document is assigned to the same cluster (lines 8 through 14). The algorithm repeats for a lower frequency $n$, as long as $n \geq 2$.

After the clustering, we can assume that if the user requests a document from the $i$-th cluster $S[i]$, he is considerably more likely to prefer a next document from $S[i]$ rather than from $S[j]$, $j \neq i$, i.e. $P = P(D^{next} \in S[i] \mid D^{prev} \in S[i], Data) \gg 1 - P$. This assumption is reasonable because by construction clusters represent densely connected (in terms of traffic) components, and the traffic across the clusters is small compared to the traffic within each cluster. In view of this observation, we broke individual user sessions down into subsessions, where each subsession consisted of documents belonging to the same cluster. The problem was thus reduced to a series of prediction problems for each cluster.

We studied the clusters by trying to find out if the documents within a cluster are topically related. We ran code previously developed at NEC Labs [4] that uses information gain to find the top features that distinguish each cluster from the rest. Table 1 shows the top features for some of the created clusters. The top features are quite consistent descriptors, suggesting that in one session a ResearchIndex user is typically interested in searching among topically-related documents.

## 4  Trigger MaxEnt

In this paper, we model $P(D^{next}|H(U), Data)$ as a maxent distribution, where $D^{next}$ is the identity of the document that will be next requested by the user $U$, given the history $H(U)$ and the available $Data$ for all other users. This choice of the maxent model is natural since our intuition is that all of the previously requested documents in the user session influence the identity of $D^{next}$. It is also clear that we cannot afford to build a high-order model, because of the sparsity and high-dimensional data, so we need to restrict ourselves to models that can be reliably estimated from the low-order statistics.

Bigrams provide one type of such statistics. In order to introduce long term dependence of $D^{next}$ on the documents that occurred in the history of the session, we define a trigger as a pair of documents $(a, b)$ in a given cluster such that $P(D^{next} = b|a \in H)$ is substantially different from $P(D^{next} = b)$. To measure the quality of triggers and in order to rank them

Table 2: Average number of hits $\bar{h}$ and height $\bar{H}$ of predictions across the clusters for different ranges of heights and using various models. The boxed numbers are the best values across all models.

| Model | | $H < 5$ | $H < 10$ | $H < 15$ | $H < 20$ | $H < 25$ |
|---|---|---|---|---|---|---|
| Mult. | $\bar{h}$ | 48.78 | 67.94 | 80.94 | 90.93 | 98.54 |
| 1 c. | $\bar{H}$ | 1.437 | 2.947 | 4.390 | 5.773 | 7.026 |
| Mult. | $\bar{h}$ | 95.49 | 120.52 | 132.07 | 138.89 | 143.33 |
| 25 c. | $\bar{H}$ | 1.421 | 2.503 | 3.312 | 3.975 | 4.528 |
| Mark. | $\bar{h}$ | 91.39 | 115.68 | 123.44 | 126.26 | 127.57 |
| 1 c. | $\bar{H}$ | 1.959 | 3.007 | 3.571 | 3.875 | 4.063 |
| Mark. | $\bar{h}$ | 89.75 | 114.49 | 122.57 | 125.61 | 127.14 |
| 25 c. | $\bar{H}$ | 1.959 | 3.047 | 3.646 | 3.972 | 4.191 |
| Maxent | $\bar{h}$ | 111.95 | 130.35 | 138.18 | 142.56 | 145.55 |
| no sm. | $\bar{H}$ | 1.510 | 2.296 | 2.858 | 3.303 | 3.694 |
| Maxent | $\bar{h}$ | 112.68 | 130.86 | 138.53 | 142.85 | 145.78 |
| w. sm. | $\bar{H}$ | 1.476 | 2.258 | 2.810 | 3.248 | 3.633 |
| Corr. | $\bar{h}$ | 111.02 | 132.87 | 140.96 | 144.99 | 147.34 |
| | $\bar{H}$ | 1.973 | 2.801 | 3.340 | 3.726 | 4.021 |

we computed mutual information between events $E_1 = \{D^{next} = b\}$ and $E_2 = \{a \in H\}$.

The set of features, together with maxent as an objective function, can be shown to lead to the following form of the conditional maxent model

$$P(D^{next}|H) \quad = \quad \frac{1}{Z(H)} exp[\sum_{s=1}^{S} \lambda_s F_s(D^{next}, H)], \tag{1}$$

where $Z(H)$ is a normalization constant ensuring that the distribution sums to 1.

The set of parameters $\{\lambda\}$ needs to be found from the following set of equations that restrict the distribution $P(D^{next}|H)$ to have the same expected value for each feature as seen in the training data:

$$\sum_{H}\sum_{D} P(D|H)F_s(D, H) \quad = \quad \sum_{H} F_s(D(H), H), \quad s = 1, \ldots, S, \tag{2}$$

where the LHS represents the expectation (up to a normalization factor) of the feature $F_s(D, H)$ with respect to the distribution $p(D|H)$ and the RHS is the actual frequency (up to the same normalization factor) of this feature in the training data. There exist efficient algorithms for finding the parameters $\{\lambda\}$ (e.g. improved iterative scaling [11]) that are known to converge if the constraints imposed on $P$ are consistent.

Under fairly general assumptions, the maxent model can also be shown to be a maximum likelihood model [11]. Employing a Gaussian prior with a zero mean on parameters $\lambda$ yields a maximum aposteriori solution that has been shown to be more accurate than the related maximum likelihood solution and other smoothing techniques for maxent models [2]. We use Gaussian smoothing in our experiments with a maxent model.

## 5 Experimental Results and Comparisons

We compared the trigger maxent model with the following models: mixture of Markov models (1 and 25 components), mixture of multinomials (1 and 25 components) and the

Table 3: Average time per 1000 predictions and average memory used by various models across 1000 clusters.

| | Time, s | Memory, KBytes |
|---|---|---|
| Mult., | 0.0049 | 0.5038 |
| Mult., 25 | 0.0559 | 12.58 |
| Markov, 1 | 0.0024 | 1.53 |
| Markov, 25 | 0.0311 | 68.23 |
| Maxent, no sm. | 0.0746 | 90.12 |
| Maxent, w. sm. | 0.0696 | 90.12 |
| Correlation | 7.2013 | 17.26 |

correlation method [1]. The definitions of the models can be found in [9]. The maxent model came in two flavors: unsmoothed and smoothed with a Gaussian prior, with 0 mean and fixed variance 2. We did not optimize the adjustable parameters of the models (such as the number of components for the mixture or the variance of the prior for maxent models) or the number of clusters (1000).

We chronologically partitioned the log into roughly 8 million training requests (covering 82 days) and 2 million test requests (covering 17 days). We used the average height of predictions on the test data as a main evaluation criteria. The height of a prediction is defined as follows. Assuming that the probability estimates $P(D|H)$ are available from a model $P$ for a fixed history $H$ and all possible values of $D$, we first sort them in the descending order of $P$ and then find the distance in terms of the number of documents to the actually requested $D$ (which we know from the test data) from the top of this sorted list. The height tells us how deep into the list the user must go in order to see the document that actually interests him. The height of a perfect prediction is 0, the maximum (worst) height for a given cluster equals the number of documents in this cluster. Since heights greater than 20 are of little practical interest, we binned the heights of predictions for each cluster. For binning purposes we used height ranges $[5K, 5(K+1))$ for $K = 0, \ldots, 4$. Within each bin we also computed the average height of predictions. Thus, the best performing model would place most of the predictions inside the bin(s) with low value(s) of $K$ and within those bins the averages would be as low as possible.

Table 2 reports the average number of hits each model makes on average in each of the bins, as well as the average height of predictions within the bin. The smoothed maxent model has the best average height of predictions across the bins and scores roughly the same number of hits in each of the bins as the correlation method. The mixture of Markov models with 25 components evidently overfits on the training data and fails to outperform a 1 component mixture. The mixture of multinomials is quite close in quality to, but still not as good as, the maxent model with respect to both the number of hits and the height predictions in each of the bins.

In Table 3, we present comparison of various models with respect to the average time taken and memory required to make a prediction. The table clearly illustrates that the maxent model (i.e., the model-based approach) is substantially more time efficient than the correlation (i.e., the memory-based approach), even despite the fact that the model takes on average more memory. In particular, our maxent approach is roughly two orders of magnitude faster than the correlation.

# 6 Conclusions and Future Work

We have described a maxent approach to generating document recommendations in ResearchIndex. We addressed the problem of sparse, high-dimensional data by introducing a clustering of the documents based on the user navigation patterns. A particular advantage of our clustering is that by its definition the traffic across the clusters is small compared to the traffic within the cluster. This advantage allowed us to decompose the original prediction problem into a set of problems corresponding to the clusters. We also demonstrated that our clustering produces highly interpretable clusters: each cluster can be assigned a topical name based on the top-extracted features.

We presented a number of models that can be used to solve a document prediction problem within cluster. We showed that the maxent model that combines zero and first order Markov terms as well as the triggers with high information content provides the best average out-of-sample performance. Gaussian smoothing improved results even further.

There are several important directions to extend the work described in this paper. First, we plan to perform "live" testing of the clustering approach and various models in ResearchIndex. Secondly, our recent work [10] suggests that for difficult prediction problems improvement beyond the plain maxent models can be sought by employing the mixtures of maxent models. We also plan to look at different clustering methods for documents (e.g., based on the content or the link structure) and try to combine prediction results for different clusterings. Our expectation is that such combining could yield better accuracy at the expense of longer running times. Finally, one could think of a (quite involved) EM algorithm that performs the clustering of the documents in a manner that would make prediction within resulting clusters easier.

**Acknowledgements** We would like to thank Steve Lawrence for making available the ResearchIndex log data, Eric Glover for running his naming code on our clusters, Kostas Tsioutsiouliklis and Darya Chudova for many useful discussions, and the anonymous reviewers for helpful suggestions.

## Footnotes

[1] http://www.researchindex.com

# References

[1] J. Breese, D. Heckerman, and C. Kadie. Empirical analysis of predictive algorithms for collaborative filtering. In *Proceedings of UAI-1998*, pages 43–52. San Francisco, CA: Morgan Kaufmann Publishers, 1998.

[2] S. Chen and R. Rosenfeld. A Gaussian prior for smoothing maximum entropy models. Technical Report CMUCS -99-108, Carnegie Mellon University, 1999.

[3] D. Cosley, S. Lawrence, and D. Pennock. An open framework for practical testing of recommender systems using ResearchIindex. In *International Conference on Very Large Databases (VLDB'02)*, 2002.

[4] E. Glover, D. Pennock, S. Lawrence, and R. Krovetz. Inferring hierarchical descriptions. Technical Report NECI TR 2002-035, NEC Research Institute, 2002.

[5] J. Goodman. Classes for fast maximum entropy training. In *Proceedings of IEEE International Conference on Acoustics, Speech, and Signal Processing*, 2001.

[6] D. Heckerman, D. Chickering, C. Meek, R. Rounthwaite, and C. Kadie. Dependency networks for density estimation, collaborative filtering, and data visualization. *Journal of Machine Learning Research*, 1:49—75, 2000.

[7] T. Hofmann and J. Puzicha. Latent class models for collaborative filtering. In *Proceedings of the Sixteenth International Joint Conference on Artificial Intelligence*, pages 688–693, 1999.

[8] S. Lawrence, C. L. Giles, and K. Bollacker. Digital libraries and Autonomous Citation Indexing. *IEEE Computer*, 32(6):67–71, 1999.

[9] D. Pavlov and D. Pennock. A maximum entropy approach to collaborative filtering in dynamic, sparse, high-dimensional domains. Technical Report NECI TR, NEC Research Institute, 2002.

[10] D. Pavlov, A. Popescul, D. Pennock, and L. Ungar. Mixtures of conditional maximum entropy models. Technical Report NECI TR, NEC Research Institute, 2002.

[11] S. D. Pietra, V. D. Pietra, and J. Lafferty. Inducing features of random fields. *IEEE Transactions on Pattern Analysis and Machine Intelligence*, 19(4):380–393, April 1997.

[12] A. Popescul, L. Ungar, D. Pennock, and S. Lawrence. Probabilistic models for unified collaborative and content-based recommendation in sparse-data environments. In *Proceedings of the Seventeenth Conference on Uncertainty in Artificial Intelligence*, pages 437–444, 2001.

[13] P. Resnick, N. Iacovou, M. Suchak, P. Bergstorm, and J. Riedl. GroupLens: An Open Architecture for Collaborative Filtering of Netnews. In *Proceedings of ACM 1994 Conference on Computer Supported Cooperative Work*, pages 175–186, Chapel Hill, North Carolina, 1994. ACM.

[14] B. Sarwar, G. Karypis, J. Konstan, and J. Riedl. Analysis of recommender algorithms for e-commerce. In *Proceedings of the 2nd ACM Conference on Electronic Commerce*, pages 158–167, 2000.

[15] G. Shani, R. Brafman, and D. Heckerman. An MDP-based recommender system. In *Proceedings of the Seventeenth Conference on Uncertainty in Artificial Intelligence*, pages 453—460, 2002.

[16] U. Shardanand and P. Maes. Social information filtering: Algorithms for automating 'word of mouth". In *Proceedings of ACM CHI'95 Conference on Human Factors in Computing Systems*, volume 1, pages 210–217, 1995.
